# Implicit Surfaces with Globally Regularised and Compactly Supported Basis Functions

**Christian Walder**[†⋆]**, Bernhard Schölkopf**[†] **& Olivier Chapelle**[†]

[†] Max Planck Institute for Biological Cybernetics, 72076 Tübingen, Germany
[⋆] The University of Queensland, Brisbane, Queensland 4072, Australia
`first.last@tuebingen.mpg.de`

## Abstract

We consider the problem of constructing a function whose zero set is to represent a surface, given sample points with surface normal vectors. The contributions include a novel means of regularising multi-scale compactly supported basis functions that leads to the desirable properties previously only associated with fully supported bases, and show equivalence to a Gaussian process with modified covariance function. We also provide a regularisation framework for simpler and more direct treatment of surface normals, along with a corresponding generalisation of the representer theorem. We demonstrate the techniques on 3D problems of up to 14 million data points, as well as 4D time series data.

## 1 Introduction

The problem of reconstructing a surface from a set of points frequently arises in computer graphics. Numerous methods of sampling physical surfaces are now available, including laser scanners, optical triangulation systems and mechanical probing methods. Inferring a surface from millions of points sampled with noise is a non-trivial task however, for which a variety of methods have been proposed. The class of *implicit* or *level set* surface representations is a rather large one, however other methods have also been suggested – for a review see [1]. The implicit surface methods closest to the present work are those that construct the implicit using regularised function approximation [2], such as the "Variational Implicits" of Turk and O'Brien [3], which produce excellent results, but at a cubic computational fitting cost in the number of points. The effectiveness of this type of approach is undisputed however, and has led researchers to look for ways to overcome the computational problems. Two main options have emerged.

The first approach uses compactly supported kernel functions (we define and discuss kernel functions in Section 2), leading to fast algorithms that are easy to implement [4]. Unfortunately however these methods are suitable for benign data sets only. As noted in [5], compactly supported basis functions "yield surfaces with many undesirable artifacts in addition to the lack of extrapolation across holes". A similar conclusion was reached in [6] which states that local processing methods are "more sensitive to the quality of input data [than] approximation and interpolation techniques based on globally-supported radial basis functions" – a conclusion corroborated by the results within a different paper from the same group [7]. The second means of overcoming the aforementioned computational problem does not suffer from these problems however, as demonstrated by the FastRBF[TM]algorithm [5], which uses the the Fast Multipole Method (FMM) [8] to overcome the computational problems of non-compactly supported kernels. The resulting method is non-trivial to implement however and to date exists only in the proprietary FastRBF[TM]package.

We believe that by applying them in a different manner, compactly supported basis functions can lead to high quality results, and the present work is an attempt to bring the reader to the same conclusion. In Section 3 we introduce a new technique for regularising such basis functions which

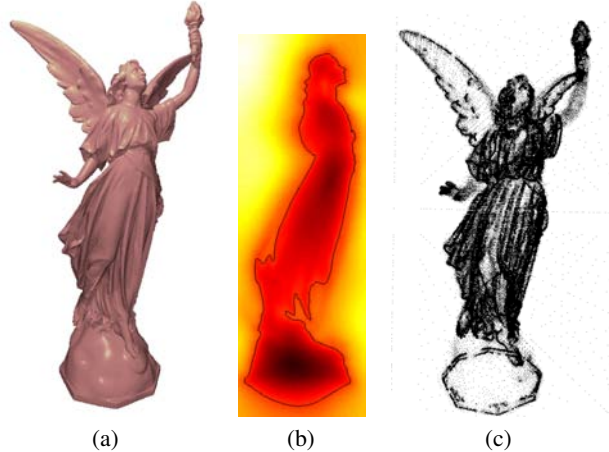

Figure 1: **(a)** Rendered implicit surface model of "Lucy", constructed from 14 million points with normals. **(b)** A planar slice that cuts the nose – the colour represents the value of the embedding function and the black line its zero level. **(c)** A black dot at each of the 364,982 compactly supported basis function centres which, along with the corresponding dilations and magnitudes, define the implicit.

(a)&emsp;&emsp;&emsp;&emsp;(b)&emsp;&emsp;&emsp;&emsp;(c)

allows high quality, highly scalable algorithms that are relatively easy to implement. We also show that the approximation can be interpreted as a Gaussian process with modified covariance function. Before doing so however, we present in Section 2 the other main contribution of the present work, which is to show how surface normal vectors can be incorporated directly into the regularised regression framework that is typically used for fitting implicit surfaces, thereby avoiding the problematic approach of constructing "off-surface" points for the regression problem. To demonstrate the effectiveness of the method we apply it to various problems in Section 4 before summarising in the final Section 5.

## 2&emsp;Implicit Surface Fitting by Regularised Regression

Here we discuss the use of regularised regression [2] for the problem of implicit surface fitting. In Section 2.1 we motivate and introduce a clean and direct means of making use of normal vectors. The following Section 2.2 extends on the ideas in Section 2.1 by formally generalising the important *representer theorem*. The final Section 2.3 discusses the choice of regulariser (and associated kernel function), as well as the associated computational problems that we overcome in Section 3.

### 2.1&emsp;Regression Based Approaches and the Use of Normal Vectors

Typically implicit surface has been done by solving a regularised regression problem [5, 4]

$$\arg \min_f \|f\|^2_{\mathcal{H}} + C \sum_{i=1}^m \left(f\left(\boldsymbol{x}_i\right) - y_i\right)^2, \tag{1}$$

where the $y_i$ are some estimate of the signed distance function at the $\boldsymbol{x}_i$, and $f$ is the embedding function which takes on the value zero on the implicit surface. The norm $\|f\|_{\mathcal{H}}$ is a *regulariser* which takes on larger values for less "smooth" functions. We take $\mathcal{H}$ to be a reproducing kernel Hilbert space (RKHS) with representer of evaluation (kernel function) $k(\cdot, \cdot)$, so that we have the *reproducing* property, $f(\boldsymbol{x}) = \langle f, k(\boldsymbol{x}, \cdot)\rangle_{\mathcal{H}}$. The solution to this problem has the form

$$f\left(\boldsymbol{x}\right) = \sum_i^m \alpha_i k\left(\boldsymbol{x}_i, \boldsymbol{x}\right). \tag{2}$$

Note as a technical aside that the thin-plate kernel case – which we will adopt – requires a somewhat more technical interpretatiosn, as it is only *conditionally* positive definite. We discuss the positive definite case for clarity only, as it is simpler and yet sufficient to demonstrate the ideas involved.

Choosing the $(\boldsymbol{x}_i, y_i)$ pairs for (2) is itself a non-trivial problem, and heuristics are typically used to prevent contradictory target values (see *e.g.* [5]). We now propose more direct method, novel in the context of implicit fitting, which avoids these problems. The approach is suggested by the fact that the normal direction of the implicit surface is given by the gradient of the embedding function

– thus normal vectors can be incorporated by regression with gradient targets. The function that we seek is the minimiser of:

$$\|f\|_{\mathcal{H}}^2 + C_1 \sum_{i=1}^{m} \left(f\left(\boldsymbol{x}_i\right)\right)^2 + C_2 \sum_{i=1}^{m} \left\|(\nabla f)\left(\boldsymbol{x}_i\right) - \boldsymbol{n}_i\right\|_{\mathbb{R}^d}^2, \tag{3}$$

which uses the given surface point/normal pairs $(\boldsymbol{x}_i, \boldsymbol{n}_i)$ directly. By imposing stationarity and using the reproducing property we can solve for the optimal $f$. A detailed derivation of this procedure is given in [1]. Here we provide only the result, which is that we have to solve for $m$ coefficients $\alpha_i$ as well as a further $md$ coefficients $\beta_{lj}$ to obtain the optimal solution

$$f\left(\boldsymbol{x}\right) = \sum_{i}^{m} \alpha_i k\left(\boldsymbol{x}_i, \boldsymbol{x}\right) + \sum_{i}^{m} \sum_{l}^{d} \beta_{li} k_l\left(\boldsymbol{x}_i, \boldsymbol{x}\right), \tag{4}$$

where we define $k_l\left(\boldsymbol{x}_i, \boldsymbol{x}\right) \doteq \left[(\nabla k)\left(\boldsymbol{x}_i, \boldsymbol{x}\right)\right]_l$, the partial derivative of $k$ in the $l$-th component of its first argument.[1] The coefficients $\boldsymbol{\alpha}$ and $\boldsymbol{\beta_l}$ of the solution are found by solving the system given by

$$\boldsymbol{0} \quad = \quad (K + I/C_1)\boldsymbol{\alpha} + \sum_{l} K_l \boldsymbol{\beta}_l \tag{5}$$

$$N_m \quad = \quad K_m \boldsymbol{\alpha} + (K_{mm} + I/C_2)\boldsymbol{\beta_k} + \sum_{l \neq m} K_{lm} \boldsymbol{\beta_l}, \quad m = 1 \ldots d \tag{6}$$

where, writing $k_{lm}$ for the second derivatives of $k(\cdot, \cdot)$ (defined similarly to the first), we've defined

$$\begin{array}{ll} \left[N_l\right]_i = \left[\boldsymbol{n}_i\right]_l; & \left[\boldsymbol{\alpha}\right]_i = \alpha_i \\ \left[\boldsymbol{\beta}_l\right]_i = \beta_{li}; & \left[K\right]_{i,j} = k(\boldsymbol{x}_i, \boldsymbol{x}_j) \\ \left[K_l\right]_{i,j} = k_l(\boldsymbol{x}_i, \boldsymbol{x}_j); & \left[K_{lm}\right]_{i,j} = k_{lm}(\boldsymbol{x}_i, \boldsymbol{x}_j). \end{array}$$

In summary, minimum norm approximation in an RKHS with gradient target values is optimally solved by a function in the span of the kernels and derivatives thereof as per Equation 4 (*cf.* Equation 2), and the coefficients of the solution are given by Equations (5) and (6). It turns out, however, that we can make a more general statement, which we do briefly in the next sub-Section.

## 2.2 The Representer Theorem with Linear Operators

The representer theorem, much celebrated in the Machine Learning community, says that the function minimising an RKHS norm along with some penalties associated with the function value at various points (as in Equation 1 for example) is a sum of kernel functions at those points (as in Equation 2). As we saw in the previous section however, if gradients also appear in the risk function to be minimised, then gradients of the kernel function appear in the optimal solution. We now make a more general statement – the case in the previous section corresponds to the following if we choose the linear operators $L_i$ (which we define shortly) as either identities or partial derivatives. The theorem is a generalisation of [9] (using the same proof idea) with equivalence if we choose all $L_i$ to be identity operators. The case of general linear operators was in fact dealt with already in [2] (which merely states the earlier result in [10]) – but only for the case of a specific loss function $c$. The following theorem therefore combines the two frameworks:

**Theorem 1**
*Denote by $\mathcal{X}$ a non-empty set, by $k$ a reproducing kernel with reproducing kernel Hilbert space $\mathcal{H}$, by $\Omega$ a strictly monotonic increasing real-valued function on $[0, \infty)$, by $c : \mathbb{R}^m \to \mathbb{R} \cup \{\infty\}$ an arbitrary cost function, and by $L_1, \ldots L_m$ a set of linear operators $\mathcal{H} \to \mathcal{H}$. Each minimiser $f \in \mathcal{H}$ of the regularised risk functional*

$$c((L_1 f)(\boldsymbol{x}_1), \ldots (L_m f)(\boldsymbol{x}_m)) + \Omega(||f||_{\mathcal{H}}^2) \tag{7}$$

*admits the form*

$$f = \sum_{i=1}^{m} \alpha_i L_i^* k_{\boldsymbol{x}_i}, \tag{8}$$

*where $k_{\boldsymbol{x}} \triangleq k(\cdot, \boldsymbol{x})$ and $L_i^*$ denotes the adjoint of $L_i$.*

*Proof.* Decompose $f$ into $f = \sum_{i=1}^{m} \alpha_i L_i^* k_{\boldsymbol{x}_i} + f_\perp$, with $\alpha_i \in \mathbb{R}$ and $\langle f_\perp, L_i^* k_{\boldsymbol{x}_i} \rangle_{\mathcal{H}} = 0$, for each $i = 1 \ldots m$. Due to the reproducing property we can write, for $j = 1 \ldots m$,

$$
\begin{aligned}
(L_j f)(\boldsymbol{x}_j) &= \langle (L_j f), k(\cdot, \boldsymbol{x}_j) \rangle_{\mathcal{H}} \\
&= \sum_{i=1}^{m} \alpha_i \langle L_j L_i^* k_{\boldsymbol{x}_i}, k(\cdot, \boldsymbol{x}_j) \rangle_{\mathcal{H}} + \langle (L_j f_\perp), k(\cdot, \boldsymbol{x}_j) \rangle_{\mathcal{H}} \\
&= \sum_{i=1}^{m} \alpha_i \langle L_j L_i^* k_{\boldsymbol{x}_i}, k(\cdot, \boldsymbol{x}_j) \rangle_{\mathcal{H}}.
\end{aligned}
$$

Thus, the first term in Equation 7 is independent of $f_\perp$. Moreover, it is clear due to orthogonality that if $f_\perp \neq 0$ then

$$
\Omega \left( \left\| \sum_{i=1}^{m} \alpha_i L_i^* k_{\boldsymbol{x}_i} + f_\perp \right\|_{\mathcal{H}}^2 \right) > \Omega \left( \left\| \sum_{i=1}^{m} \alpha_i L_i^* k_{\boldsymbol{x}_i} \right\|_{\mathcal{H}}^2 \right),
$$

so that for any fixed $\alpha_i \in \mathbb{R}$, Equation 7 is minimised when $f_\perp = 0$. $\qquad \square$

## 2.3 Thin Plate Regulariser and Associated Kernel

As is well known (see *e.g.* [2]), the choice of regulariser (the function norm in Equation 3) leads to a particular kernel function $k(\cdot, \cdot)$ to be used in Equation 4. For geometrical problems, an excellent regulariser is the *thin-plate* energy, which for arbitrary order $m$ and dimension $d$ is given by [2]:

$$
\|f\|_{\mathcal{H}}^2 = \langle \psi f, \psi f \rangle_{L_2} \tag{9}
$$

$$
= \sum_{i_1=1}^{d} \cdots \sum_{i_m=1}^{d} \int_{x_1=-\infty}^{\infty} \cdots \int_{x_d=-\infty}^{\infty} \left( \frac{\partial}{\partial x_{i_1}} \cdots \frac{\partial}{\partial x_{i_m}} f \right) \left( \frac{\partial}{\partial x_{i_1}} \cdots \frac{\partial}{\partial x_{i_m}} f \right) \mathrm{d}x_1 \ldots \mathrm{d}x_d, \tag{10}
$$

where $\psi$ is a regularisation operator taking all partial derivatives of order $m$, which corresponds to a "radial" kernel function of the form $k(\boldsymbol{x}, \boldsymbol{y}) = t(\|\boldsymbol{x} - \boldsymbol{y}\|)$, where [11]

$$
t(r) = \begin{cases} r^{2m-d} \ln(r) & \text{if } 2m > d \text{ and } d \text{ is even,} \\ r^{2m-d} & \text{otherwise.} \end{cases}
$$

There are a number of good reasons to use this regulariser rather than those leading to compactly supported kernels, as we touched on in the introduction. The main problem with compactly supported kernels is that the corresponding regularisers are somewhat poor for geometrical problems – they always draw the function towards some nominal constant as one moves away from the data, thereby implementing the non-intuitive behaviour of regularising the constant function and making interpolation impossible – for further discussion see [1] as well as [5, 6, 7]. The scheme we propose in Section 3 solves these problems, previously associated with compactly supported basis functions, by defining and computing the regulariser separately from the function basis.

# 3 A Fast Scheme using Compactly Supported Basis Functions

Here we present a fast approximate scheme for solving the problem of the previous Section, in which we restrict the class of functions to the span of a compactly supported, multi-scale basis, as described in Section 3.1, and minimise the thin-plate regulariser within this span as per Section 3.2.

## 3.1 Restricting the Set of Available Functions

Computationally, using the thin-plate spline leads to the problem that the linear system we need to solve (Equations 5 and 6), which is of size $m(d + 1)$, is dense in the sense of having almost all non-zero entries. Since solving such a system naïvely has a cubic time complexity in $m$, we propose forcing $f(\cdot)$ to take the form:

$$
f(\cdot) = \sum_{k=1}^{p} \pi_k f_k(\cdot), \tag{11}
$$

where the individual basis functions are $f_k(\cdot) = \phi(\|\boldsymbol{v}_k - \cdot\| / s_k)$ for some function $\phi : \mathbb{R}^+ \to \mathbb{R}$ with support $[0, 1)$. The $\boldsymbol{v}_k$ and $s_k$ are the basis function centres and dilations (or scales), respectively. For $\phi$ we choose the $B_3$-spline function:

$$\phi(r) = \sum_{n=0}^{4} \frac{(-1)^n}{d!} \binom{n}{d+1} (r + (\frac{d+1}{2} - n))_+^d, \tag{12}$$

although this choice is rather inconsequential since, as we shall ensure, the regulariser is unrelated to the function basis – any smooth compactly supported basis function could be used. In order to achieve the same interpolating properties as the thin-plate spline, we wish to minimise our regularised risk function given by Equation 3 within the span of Equation 11. The key to doing this is to note that as given before in Equation 9, the regulariser (function norm) can be written as $\|f\|_{\mathcal{H}}^2 = \langle \psi f, \psi f \rangle_{L_2}$. Given this fact, a straightforward calculation leads to the following system for the optimal $\pi_k$ (in the sense of minimising Equation 3):

$$\left( K_{\text{reg}} + C_1 K_{xv}^{\text{T}} K_{xv} + C_2 \sum_{l=1}^{d} K_{xvl}^{\text{T}} K_{xvl} \right) \boldsymbol{\pi} = C_2 \sum_{l=1}^{d} K_{xvl} N_l, \tag{13}$$

where we have defined the following matrices:

$$[K_{\text{reg}}]_{k,k'} = \langle \psi f_k, \psi f_{k'} \rangle_{L_2}; \quad [K_{xv}]_{i,k} = f_k(\boldsymbol{x}_i); \quad [K_{xvl}]_{i,k} = [(\nabla f_k)(\boldsymbol{x}_i)]_l;$$

$$[\boldsymbol{\pi}]_k = \pi_k; \quad [N_l]_i = [\boldsymbol{n}_i]_l.$$

The computational advantage is that the coefficients that we need are now given by a *sparse p-dimensional* positive semi-definite linear system, which can be constructed efficiently by simple code that takes advantage of software libraries for fast nearest neighbour type searches (see *e.g.* [12]). The system can then be solved efficiently using *conjugate gradient* type methods. In [1] we describe how we construct a basis with $p \ll m$ that results in a highly sparse linear system, but that still contains good solutions. The critical matter of computing $K_{\text{reg}}$ is dealt with next.

### 3.2 Computing the Regularisation Matrix

We now come to the crucial point of calculating $K_{\text{reg}}$, which can be thought of as the regularisation matrix. The present Section is highly related to [13], however there numerical methods were resorted to for the calculation of $K_{\text{reg}}$ – presently we shall derive closed form solutions. Also worth comparing to the present Section is [14], where a prior over the expansion coefficients (here the $\boldsymbol{\pi}$) is used to mimic a given regulariser within an arbitrary basis, achieving a similar result but without the computational advantages we are aiming for. As we have already noted we can write $\|f\|_{\mathcal{H}}^2 = \langle \psi f, \psi f \rangle_{L_2}$ [2], so that for the function given by Equation 11 we have:

$$\left\| \sum_{j=1}^{p} \pi_j f_j(\cdot) \right\|_{\mathcal{H}}^2 = \left\langle \psi \sum_{j=1}^{p} \pi_j f_j(\cdot), \psi \sum_{k=1}^{p} \pi_k f_k(\cdot) \right\rangle_{L_2}$$

$$= \sum_{j,k=1}^{p} \pi_j \pi_k \langle \psi f_j(\cdot), \psi f_k(\cdot) \rangle_{L_2} \doteq \boldsymbol{\pi}^{\text{T}} K_{\text{reg}} \boldsymbol{\pi}.$$

To build the sparse matrix $K_{\text{reg}}$, a fast range search library (*e.g.* [12]) can be used to identify the non-zero entries – that is, all those $[K_{\text{reg}}]_{i,j}$ for which $i$ and $j$ satisfy $\|\boldsymbol{v}_i - \boldsymbol{v}_j\| \leq s_i + s_j$. In order to evaluate $\langle \psi f_j(\cdot), \psi f_k(\cdot) \rangle_{L_2}$, it is necessary to solve the integral of Equation 10, the full derivation of which we relegate to [1] – here we just provide the main results. It turns out that since the $f_i$ are all dilations and translations of the same function $\phi(\|\cdot\|)$, then it is sufficient solve for the following function of $s_i$, $s_j$ and $\boldsymbol{d} \doteq \boldsymbol{v}_i - \boldsymbol{v}_j$:

$$\langle \psi \phi((\cdot) s_i - \boldsymbol{d}), \psi \phi((\cdot) s_j) \rangle_{L_2},$$

which it turns out is given by

$$\mathcal{F}_{\boldsymbol{\omega}}^{-1} \left[ \frac{(2\pi j \|\boldsymbol{\omega}\|)^{2m}}{|s_1 s_2|} \Phi(\frac{\boldsymbol{\omega}}{s_1}) \Phi(\frac{\boldsymbol{\omega}}{s_2}) \right] (\boldsymbol{d}), \tag{14}$$

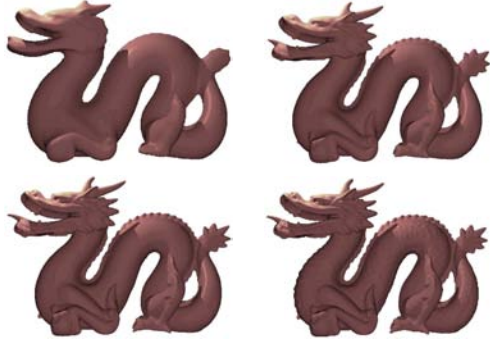
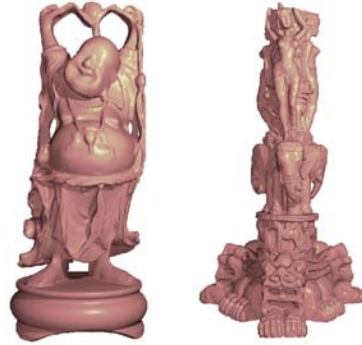

Figure 2: Various values of the regularisation parameters lead to various amounts of "smoothing" – here we set $C_1 = C_2$ in Equation 3 to an increasing value from top-left to bottom-right of the figure.

Figure 3: Ray traced three dimensional implicits, "Happy Buddha" (543K points with normals) and the "Thai Statue" (5 million points with normals).

where $j^2 = -1$, $\Phi = \mathcal{F}_x[\phi(\boldsymbol{x})]$, and by $\mathcal{F}$ (and $\mathcal{F}^{-1}$) we mean the Fourier (inverse Fourier) transform operators in the subscripted variable. Computing Fourier transforms in $d$ dimensions difficult in general, but for radial functions $g(\boldsymbol{x}) = g_r(\|\boldsymbol{x}\|)$ it may be made easier by the fact that the Fourier transform in $d$ dimensions (as well as its inverse) can be computed by the single integral:

$$\mathcal{F}_{\boldsymbol{x}}\left[g_r(\|\boldsymbol{x}\|)\right](\|\boldsymbol{\omega}\|) = \frac{(2\pi)^{\frac{d}{2}}}{\|\boldsymbol{\omega}\|^{\frac{d-2}{2}}} \int_0^\infty r^{\frac{d}{2}} g_r(r) J_{\frac{d-2}{2}}(\|\boldsymbol{\omega}\|r)\mathrm{d}r,$$

where $J_\nu(r)$ is the $\nu$-th order Bessel function of the first kind.

Unfortunately the integrals required to attain Equation 14 in closed form cannot be solved for general dimensionality $d$, regularisation operator $\psi$ and basis function form $\phi$, however we did manage to solve them for arguably the most useful case: $d = 3$ with the $m = 2$ thin plate energy and the $B_3$-spline basis function of Equation 12. The resulting expressions are rather unwieldy however, so we give only an implementation in the C language in the Appendix of [1], where we also show that for the cases that cannot be solved analytically the required integral can at worst always be transformed to a two dimensional integral for which one can use numerical methods.

## 3.3 Interpretation as a Gaussian Process

Presently we use ideas from [15] to demonstrate that the approximation described in this Section 3 is equivalent to inference in an exact Gaussian Process with covariance function depending on the choice of function basis. Placing a multivariate Gaussian prior over the coefficients in (11), namely $\boldsymbol{\pi} \sim \mathcal{N}(\mathbf{0}, K_{\mathrm{reg}}^{-1})$, we see that $f$ obeys a zero mean Gaussian process prior – writing $[f_x]_i = f(\boldsymbol{x}_i)$ and denoting expectations by $E[\cdot]$ we have for the covariance

$$
\begin{aligned}
E\left[f_x f_x^{\mathrm{T}}\right] &= K_{xz} E\left[\boldsymbol{\pi}\boldsymbol{\pi}^{\mathrm{T}}\right] K_{xz}^{\mathrm{T}} \\
&= K_{xz} K_{\mathrm{reg}}^{-1} K_{xz}^{\mathrm{T}}
\end{aligned}
$$

Now, assuming an *iid* Gaussian noise model with variance $\sigma^2$ and defining $K_{xt}$ *etc.* similarly to $K_{xz}$ we can immediately write the joint distribution between the observation at a test point $\boldsymbol{t}$, that is $y_t \sim \mathcal{N}\left(f(\boldsymbol{t}), \sigma^2\right)$ and the vector of observations at the $\boldsymbol{x}_i$, namely $\boldsymbol{y}_x \sim \mathcal{N}\left(f_x, \sigma^2 I\right)$, which is

$$p(\boldsymbol{y}_x, y_t) = \mathcal{N}\left(\mathbf{0}, \begin{pmatrix} \left(K_{xz}K_{\mathrm{reg}}^{-1}K_{zx} + \sigma^2 I\right) & K_{xz}K_{\mathrm{reg}}^{-1}K_{zt} \\ K_{tz}K_{\mathrm{reg}}^{-1}K_{zx} & \left(K_{tz}K_{\mathrm{reg}}^{-1}K_{zt} + \sigma^2 I\right) \end{pmatrix}\right).$$

The posterior distribution is therefore itself Gaussian, $p(y_t|\boldsymbol{y}_x) \sim \mathcal{N}\left(\mu_{y_t|\boldsymbol{y}_x}, \Sigma_{y_t|\boldsymbol{y}_x}\right)$, and we can employ a well known expression[2] for the marginals of a multivariate Gaussian followed by the Matrix inversion lemma to derive an expression for the mean of the posterior,

$$
\begin{aligned}
\mu_{\boldsymbol{t}|\boldsymbol{y}} &= \left(K_{xz}K_{\mathrm{reg}}^{-1}K_{zt}\right)^{\mathrm{T}}\left(K_{xz}K_{\mathrm{reg}}^{-1}K_{zx} + \sigma^2 I\right)^{-1}\boldsymbol{y} \\
&= K_{tz}\left(\sigma^2 K_{\mathrm{reg}} + K_{xz}^{\mathrm{T}}K_{xz}\right)^{-1}K_{xz}^{\mathrm{T}}\boldsymbol{y}.
\end{aligned}
$$

| Name | # Points | # Bases | Basis | $K_{\text{reg}}$ | $K_{xv}$, $K_{zv\nabla}$ | Multiply | Solve | Total |
|---|---|---|---|---|---|---|---|---|
| Bunny | 34834 | 9283 | 0.4 | 2.4 | 3.7 | 11.7 | 20.4 | 38.7 |
| Face | 75970 | 7593 | 0.7 | 1.9 | 7.0 | 20.3 | 16.0 | 46.0 |
| Armadillo | 172974 | 45704 | 6.6 | 8.5 | 37.0 | 123.4 | 72.3 | 247.9 |
| Dragon | 437645 | 65288 | 14.4 | 16.3 | 70.9 | 322.8 | 1381.4 | 1805.7 |
| Buddha | 543197 | 105993 | 117.4 | 27.4 | 99.4 | 423.7 | 2909.3 | 3577.2 |
| Asian Dragon | 3609455 | 232197 | 441.6 | 60.9 | 608.3 | 1885.0 | 1009.5 | 4005.2 |
| Thai Statue | 4999996 | 530966 | 3742.0 | 197.5 | 1575.6 | 3121.2 | 2569.5 | 11205.7 |
| Lucy | 14027872 | 364982 | 1425.8 | 170.5 | 3484.1 | 9367.7 | 1340.5 | 15788.5 |

Table 1: Timing results with a 2.4GHz AMD Opteron 850 processor, for various 3D data sets. Column one is the number of points, each of which has an associated normal vector, and column two is the number of basis vectors (the $p$ of Section 3.1). The remaining columns are all in units of seconds: column three is the time taken to construct the function basis, columns four and five are the times required to construct the indicated matrices, column six is the time required to multiply the matrices as per Equation 13, column seven is the time required to solve that same equation for $\boldsymbol{\pi}$ and the final column is the total fitting time.

By comparison with (11) and (13) (but with $C_1 = 1/\sigma^2$, $C_2 = 0$ and $\boldsymbol{y} = \boldsymbol{0}$) we can see that the mean of the posterior distribution is identical to our approximate regularised solution based on compactly supported basis functions. For the corresponding posterior variance we have

$$
\begin{aligned}
\Sigma_{y_t|\boldsymbol{y}_x} &= \left(K_{tz}K_{\text{reg}}^{-1}K_{zt} + \sigma^2\right) - \left(K_{tz}K_{\text{reg}}^{-1}K_{zx}\right)\left(K_{xz}K_{\text{reg}}^{-1}K_{zx} + \sigma^2 I\right)^{-1}\left(K_{xz}K_{\text{reg}}^{-1}K_{zt}\right) \\
&= \sigma^2 K_{tz}\left(\sigma^2 K_{\text{reg}} + K_{xz}^{\mathrm{T}}K_{xz}\right)^{-1}K_{zt} + \sigma^2.
\end{aligned}
$$

## 4 Experiments

We fit models to 3D data sets of up to 14 million data points – timings are given in Table 1, where we also see that good compression ratios are attained, in that relatively few basis functions represent the shapes. Also note that the fitting time scales rather well, from 38 seconds for the Stanford Bunny (35 thousand points with normals) to 4 hours 23 minutes for the Lucy statue (14 million points with normals $\approx 14 \times 10^6 \times (1$ value term $+ 3$ gradient terms $) \approx 56$ million "regression targets"). Taking account of the different hardware the times seem to be similar to those of the FMM approach [5].

Some rendered examples are given in Figures 1 and 3, and the well-behaved nature of the implicit over the entire 3D volume of interest is shown for the Lucy data-set in the accompanying video. In practice the system is extremely robust and produces excellent results without any parameter adjustment – smaller values of $C_1$ and $C_2$ in Equation 3 simply lead to the smoothing effect shown in Figure 2. The system also handles missing and noisy data gracefully, as demonstrated in [1].

Higher dimensional implicit surfaces are also possible, interesting being a 4D representation (3D + "time") of a moving 3D shape – one use for this being the construction of animation sequences from a time series of 3D point cloud data – in this case both spatial and temporal information can help to resolve noise or missing data problems within individual scans. We demonstrate this in the accompanying video, which shows that 4D surfaces yield superior 3D animation results in comparison to a sequence of 3D models. Also interesting are interpolations in 4D – in the accompanying video we effectively interpolate between two three dimensional shapes.

## 5 Summary

We have presented ideas both theoretically and practically useful for the computer graphics and machine learning communities, demonstrating them within the framework of implicit surface fitting. Many authors have demonstrated fast but limited quality results that occur with compactly supported function bases. The present work differs by precisely minimising a well justified regulariser within the span of such a basis, achieving fast *and* high quality results. We also showed how normal vectors can be incorporated directly into the usual regression based implicit surface fitting framework, giving a generalisation of the representer theorem. We demonstrated the algorithm on 3D problems of up to 14 million data points and in the accompanying video we showed the advantage of constructing a 4D surface (3D + time) for 3D animation, rather than a sequence of 3D surfaces.

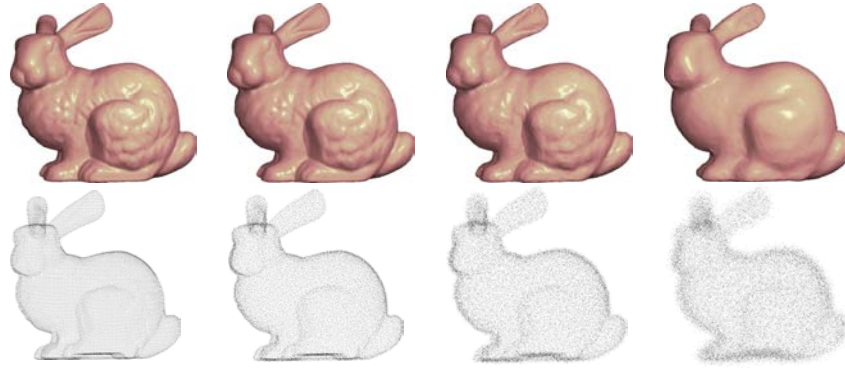

Figure 4: Reconstruction of the Stanford bunny after adding Gaussian noise with standard deviation, from left to right, 0, 0.6, 1.5 and 3.6 percent of the radius of the smallest enclosing sphere – the normal vectors were similarly corrupted assuming they had length equal to this radius. The parameters $C_1$ and $C_2$ were chosen automatically using five-fold cross validation.

## Footnotes

[1] Square brackets with subscripts indicate matrix elements: $\left[\boldsymbol{a}\right]_i$ is the $i$-th element of the vector $\boldsymbol{a}$.

[2] $\left(\begin{pmatrix} \boldsymbol{x} \\ \boldsymbol{y} \end{pmatrix} \sim \mathcal{N}\left(\begin{pmatrix} \boldsymbol{a} \\ \boldsymbol{b} \end{pmatrix}, \begin{pmatrix} A & C \\ C^{\mathrm{T}} & B \end{pmatrix}\right)\right) \Rightarrow \left(\boldsymbol{x}|\boldsymbol{y} \sim \mathcal{N}\left(a + CB^{-1}(\boldsymbol{y} - \boldsymbol{b}), A - CB^{-1}C^{\mathrm{T}}\right)\right)$

# References

[1] C. Walder, B. Schölkopf, and O. Chapelle. Implicit surface modelling with a globally regularised basis of compact support. Technical report, Max Planck Institute for Biological Cybernetics, Department of Empirical Inference, Tbingen, Germany, April 2006.

[2] G. Wahba. *Spline Models for Observational Data*. Series in Applied Mathematics, Vol. 59, SIAM, Philadelphia, 1990.

[3] Greg Turk and James F. O'Brien. Shape transformation using variational implicit functions. In *Proceedings of ACM SIGGRAPH 1999*, pages 335–342, August 1999.

[4] Bryan S. Morse, Terry S. Yoo, David T. Chen, Penny Rheingans, and K. R. Subramanian. Interpolating implicit surfaces from scattered surface data using compactly supported radial basis functions. In *SMI '01: Proc. Intl. Conf. on Shape Modeling & Applications*, Washington, 2001. IEEE Computer Society.

[5] J. C. Carr, R. K. Beatson, J. B. Cherrie, T. J. Mitchell, W. R. Fright, B. C. McCallum, and T. R. Evans. Reconstruction and representation of 3d objects with radial basis functions. In *ACM SIGGRAPH 2001*, pages 67–76. ACM Press, 2001.

[6] Yutaka Ohtake, Alexander Belyaev, Marc Alexa, Greg Turk, and Hans-Peter Seidel. Multi-level partition of unity implicits. *ACM Transactions on Graphics*, 22(3):463–470, July 2003.

[7] Y. Ohtake, A. Belyaev, and Hans-Peter Seidel. A multi-scale approach to 3d scattered data interpolation with compactly supported basis functions. In *Proc. Intl. Conf. Shape Modeling*, Washington, 2003. IEEE Computer Society.

[8] L. Greengard and V. Rokhlin. A fast algorithm for particle simulations. *J. Comp. Phys.*, pages 280–292, 1997.

[9] Bernhard Schölkopf, Ralf Herbrich, and Alex J. Smola. A generalized representer theorem. In *COLT '01/EuroCOLT '01: Proceedings of the 14th Annual Conference on Computational Learning Theory*, pages 416–426, London, UK, 2001. Springer-Verlag.

[10] G. Kimeldorf and G. Wahba. Some results on Tchebycheffian spline functions. *Journal of Mathematical Analysis and Applications*, 33:82–95, 1971.

[11] J. Duchon. Splines minimizing rotation-invariant semi-norms in sobolev spaces. *Constructive Theory of Functions of Several Variables*, pages 85–100, 1977.

[12] C. Merkwirth, U. Parlitz, and W. Lauterborn. Fast nearest neighbor searching for nonlinear signal processing. *Phys. Rev. E*, 62(2):2089–2097, 2000.

[13] Christian Walder, Olivier Chapelle, and Bernhard Schölkopf. Implicit surface modelling as an eigenvalue problem. *Proceedings of the 22nd International Conference on Machine Learning*, 2005.

[14] M. O. Franz and P. V. Gehler. How to choose the covariance for gaussian process regression independently of the basis. In *Proc. Gaussian Processes in Practice Workshop*, 2006.

[15] J. Quionero Candela and C. E. Rasmussen. A unifying view of sparse approximate gaussian process regression. *Journal of Machine Learning Research*, 6:1935–1959, 12 2005.
